# Convergent Temporal-Difference Learning with Arbitrary Smooth Function Approximation

**Hamid R. Maei**
University of Alberta
Edmonton, AB, Canada

**Csaba Szepesvári**[*]
University of Alberta
Edmonton, AB, Canada

**Shalabh Bhatnagar**
Indian Institute of Science
Bangalore, India

**Doina Precup**
McGill University
Montreal, QC, Canada

**David Silver**
University of Alberta,
Edmonton, AB, Canada

**Richard S. Sutton**
University of Alberta,
Edmonton, AB, Canada

## Abstract

We introduce the first temporal-difference learning algorithms that converge with smooth value function approximators, such as neural networks. Conventional temporal-difference (TD) methods, such as TD($\lambda$), Q-learning and Sarsa have been used successfully with function approximation in many applications. However, it is well known that off-policy sampling, as well as nonlinear function approximation, can cause these algorithms to become unstable (i.e., the parameters of the approximator may diverge). Sutton et al. (2009a, 2009b) solved the problem of off-policy learning with linear TD algorithms by introducing a new objective function, related to the Bellman error, and algorithms that perform stochastic gradient-descent on this function. These methods can be viewed as natural generalizations to previous TD methods, as they converge to the same limit points when used with linear function approximation methods. We generalize this work to nonlinear function approximation. We present a Bellman error objective function and two gradient-descent TD algorithms that optimize it. We prove the asymptotic almost-sure convergence of both algorithms, for any finite Markov decision process and any smooth value function approximator, to a locally optimal solution. The algorithms are incremental and the computational complexity per time step scales linearly with the number of parameters of the approximator. Empirical results obtained in the game of Go demonstrate the algorithms' effectiveness.

## 1 Introduction

We consider the problem of estimating the value function of a given stationary policy of a Markov Decision Process (MDP). This problem arises as a subroutine of generalized policy iteration and is generally thought to be an important step in developing algorithms that can learn good control policies in reinforcement learning (e.g., see Sutton & Barto, 1998). One widely used technique for value-function estimation is the TD($\lambda$) algorithm (Sutton, 1988). A key property of the TD($\lambda$) algorithm is that it can be combined with function approximators in order to generalize the observed data to unseen states. This generalization ability is crucial when the state space of the MDP is large or infinite (e.g., TD-Gammon, Tesauro, 1995; elevator dispatching, Crites & Barto, 1997; job-shop scheduling, Zhang & Dietterich, 1997). TD($\lambda$) is known to converge when used with linear function approximators, if states are sampled according to the policy being evaluated – a scenario called on-policy learning (Tsitsiklis & Van Roy, 1997). However, the absence of either of these requirements can cause the parameters of the function approximator to diverge when trained with TD methods (e.g., Baird, 1995; Tsitsiklis & Van Roy, 1997; Boyan & Moore, 1995). The question of whether it is possible to create TD-style algorithms that are guaranteed to converge when used with nonlinear function approximation has remained open until now. Residual gradient algorithms (Baird, 1995)

---

[*]On leave from MTA SZTAKI, Hungary.

attempt to solve this problem by performing gradient descent on the Bellman error. However, unlike TD, these algorithms usually require two independent samples from each state. Moreover, even if two samples are provided, the solution to which they converge may not be desirable (Sutton et al., 2009b provides an example).

In this paper we define the first TD algorithms that are stable when used with smooth nonlinear function approximators (such as neural networks). Our starting point is the family of TD-style algorithms introduced recently by Sutton et al. (2009a, 2009b). Their goal was to address the instability of TD learning with linear function approximation, when the policy whose value function is sought differs from the policy used to generate the samples (a scenario called off-policy learning). These algorithms were designed to approximately follow the gradient of an objective function whose unique optimum is the fixed point of the original TD(0) algorithm. Here, we extend the ideas underlying this family of algorithms to design TD-like algorithms which converge, under mild assumptions, almost surely, with smooth nonlinear approximators. Under some technical conditions, the limit points of the new algorithms correspond to the limit points of the original (not necessarily convergent) nonlinear TD algorithm. The algorithms are incremental, and the cost of each update is linear in the number of parameters of the function approximator, as in the original TD algorithm.

Our development relies on three main ideas. First, we extend the objective function of Sutton et al. (2009b), in a natural way, to the nonlinear function approximation case. Second, we use the weight-duplication trick of Sutton et al. (2009a) to derive a stochastic gradient algorithm. Third, in order to implement the parameter update efficiently, we exploit a nice idea due to Pearlmutter (1994), allowing one to compute exactly the product of a vector and a Hessian matrix in linear time. To overcome potential instability issues, we introduce a projection step in the weight update. The almost sure convergence of the algorithm then follows from standard two-time-scale stochastic approximation arguments.

In the rest of the paper, we first introduce the setting and our notation (Section 2), review previous relevant work (Section 3), introduce the algorithms (Section 4), analyze them (Section 5) and illustrate the algorithms' performance (Section 6).

## 2 Notation and Background

We consider policy evaluation in finite state and action Markov Decision Processes (MDPs).[1] An MDP is described by a 5-tuple $(\mathcal{S}, \mathcal{A}, P, r, \gamma)$, where $\mathcal{S}$ is the finite state space, $\mathcal{A}$ is the finite action space, $P = (P(s'|s,a))_{s,s' \in \mathcal{S}, a \in \mathcal{A}}$ are the transition probabilities ($P(s'|s,a) \geq 0$, $\sum_{s' \in \mathcal{S}} P(s'|s,a) = 1$, for all $s \in \mathcal{S}, a \in \mathcal{A}$), $r = (r(s,a,s'))_{s,s' \in \mathcal{S}, a \in \mathcal{A}}$ are the real-valued immediate rewards and $\gamma \in (0,1)$ is the discount factor. The policy to be evaluated is a mapping $\pi : \mathcal{S} \times \mathcal{A} \to [0,1]$. The value function of $\pi$, $V^\pi : \mathcal{S} \to \mathbb{R}$, maps each state $s$ to a number representing the infinite-horizon expected discounted return obtained if policy $\pi$ is followed from state $s$. Formally, let $s_0 = s$ and for $t \geq 0$ let $a_t \sim \pi(s_t, \cdot)$, $s_{t+1} \sim P(\cdot|s_t, a_t)$ and $r_{t+1} = r(s_t, a_t, s_{t+1})$. Then $V^\pi(s) = \mathbb{E}[\sum_{t=0}^{\infty} \gamma^t r_{t+1}]$. Let $R^\pi : S \to \mathbb{R}$, with $R^\pi(s) = \sum_{s' \in \mathcal{S}} \sum_{a \in \mathcal{A}} \pi(s,a) P(s'|s,a) r(s,a,s')$, and let $P^\pi : \mathcal{S} \times \mathcal{S} \to [0,1]$ be defined as $P^\pi(s,s') = \sum_{a \in \mathcal{A}} \pi(s,a) P(s'|s,a)$. Assuming a canonical ordering on the elements of $\mathcal{S}$, we can treat $V^\pi$ and $R^\pi$ as vectors in $\mathbb{R}^{|\mathcal{S}|}$, and $P^\pi$ as a matrix in $\mathbb{R}^{|\mathcal{S}| \times |\mathcal{S}|}$. It is well-known that $V^\pi$ satisfies the so-called *Bellman equation*:

$$V^\pi = R^\pi + \gamma P^\pi V^\pi.$$

Defining the operator $T^\pi : \mathbb{R}^{|\mathcal{S}|} \to \mathbb{R}^{|\mathcal{S}|}$ as $T^\pi V = R^\pi + \gamma P^\pi V$, the Bellman equation can be written compactly as $V^\pi = T^\pi V^\pi$. To simplify the notation, from now on we will drop the superscript $\pi$ everywhere, since the policy to be evaluated will be kept fixed.

Assume that the policy to be evaluated is followed and it gives rise to the trajectory $(s_0, a_0, r_1, s_1, a_1, r_2, s_2, \ldots)$. The problem is to estimate $V$, given a finite prefix of this trajectory. More generally, we may assume that we are given an infinite sequence of 3-tuples, $(s_k, r_k, s'_k)$, that satisfies the following:

**Assumption A1** $(s_k)_{k \geq 0}$ is an $\mathcal{S}$-valued stationary Markov process, $s_k \sim d(\cdot)$, $r_k = R(s_k)$ and $s'_k \sim P(s_k, \cdot)$.

We call $(s_k, r_k, s'_k)$ the $k^{\text{th}}$ transition. Since we assume stationarity, we will sometimes drop the index $k$ and use $(s, r, s')$ to denote a random transition. Here $d(\cdot)$ denotes the probability distribution over initial states for a transition; let $D \in \mathbb{R}^{|\mathcal{S}| \times |\mathcal{S}|}$ be the corresponding diagonal matrix. The problem is still to estimate $V$ given a finite number of transitions.

When the state space is large (or infinite) a function approximation method can be used to facilitate the generalization of observed transitions to unvisited or rarely visited states. In this paper we focus on methods that are smoothly parameterized with a finite-dimensional parameter vector $\theta \in \mathbb{R}^n$. We denote by $V_\theta(s)$ the value of state $s \in \mathcal{S}$ returned by the function approximator with parameters $\theta$. The goal of policy evaluation becomes to find $\theta$ such that $V_\theta \approx V$.

## 3   TD Algorithms with function approximation

The classical TD(0) algorithm with function approximation (Sutton, 1988; Sutton & Barto, 1998) starts with an arbitrary value of the parameters, $\theta_0$. Upon observing the $k^{\text{th}}$ transition, it computes the scalar-valued *temporal-difference error*,

$$\delta_k = r_k + \gamma V_{\theta_k}(s'_k) - V_{\theta_k}(s_k),$$

which is then used to update the parameter vector as follows:

$$\theta_{k+1} \leftarrow \theta_k + \alpha_k \, \delta_k \nabla V_{\theta_k}(s_k). \tag{1}$$

Here $\alpha_k$ is a deterministic positive step-size parameter, which is typically small, or (for the purpose of convergence analysis) is assumed to satisfy the Robbins-Monro conditions: $\sum_{k=0}^{\infty} \alpha_k = \infty$, $\sum_{k=0}^{\infty} \alpha_k^2 < \infty$. We denote by $\nabla V_\theta(s) \in \mathbb{R}^n$ the gradient of $V$ w.r.t. $\theta$ at $s$.

When the TD algorithm converges, it must converge to a parameter value where, in expectation, the parameters do not change:

$$\mathbb{E}[\delta \, \nabla V_\theta(s)] = 0, \tag{2}$$

where $s, \delta$ are random and share the common distribution underlying $(s_k, \delta_k)$; in particular, $(s, r, s')$ are drawn as in Assumption A1 and $\delta = r + \gamma V_\theta(s') - V_\theta(s)$.

However, it is well known that TD(0) may not converge; the stability of the algorithm is affected both by the actual function approximator $V_\theta$ and by the way in which transitions are sampled. Sutton et al (2009a, 2009b) tackled this problem in the case of linear function approximation, in which $V_\theta(s) = \theta^\top \phi(s)$, where $\phi : \mathcal{S} \rightarrow \mathbb{R}^n$, but where transitions may be sampled in an off-policy manner. From now on we use the shorthand notation $\phi = \phi(s)$, $\phi' = \phi(s')$.

Sutton et al. (2009b) rely on an error function, called *mean-square projected Bellman error (MSPBE)*[2], which has the same unique optimum as Equation (2). This function, which we denote $J$, projects the Bellman error measure, $TV_\theta - V_\theta$ onto the linear space $\mathcal{M} = \{V_\theta \,|\, \theta \in \mathbb{R}^n\}$ with respect to the metric $\| \cdot \|_D$. Hence, $\Pi V = \arg\min_{V' \in \mathcal{M}} \|V' - V\|_D^2$. More precisely:

$$J(\theta) = \| \Pi(TV_\theta - V_\theta) \|_D^2 = \| \Pi \, TV_\theta - V_\theta \|_D^2 = \mathbb{E}[\delta\phi]^\top \mathbb{E}[\phi\phi^\top]^{-1} \mathbb{E}[\delta\phi], \tag{3}$$

where $\|V\|_D$ is the weighted quadratic norm defined by $\|V\|_D^2 = \sum_{s \in \mathcal{S}} d(s)V(s)^2$, and the scalar TD(0) error for a given transition $(s, r, s')$ is $\delta = r + \gamma\theta^\top\phi' - \theta^\top\phi$.

The negative gradient of the MSPBE objective function is:

$$-\frac{1}{2}\nabla J(\theta) = \mathbb{E}\big[(\phi - \gamma\phi')\phi^\top w\big] = \mathbb{E}[\delta\phi] - \gamma\mathbb{E}\big[\phi'\phi^\top\big] \, w, \tag{4}$$

where $w = \mathbb{E}[\phi\phi^\top]^{-1}\mathbb{E}[\delta\phi]$. Note that $\delta$ depends on $\theta$, hence $w$ depends on $\theta$. In order to develop an efficient ($O(n)$) stochastic gradient algorithm, Sutton et al. (2009a) use a *weight-duplication trick*. They introduce a new set of weights, $w_k$, whose purpose is to estimate $w$ for a fixed value of the $\theta$ parameter. These weights are updated on a "fast" timescale, as follows:

$$w_{k+1} = w_k + \beta_k(\delta_k - \phi_k^\top w_k)\phi_k. \tag{5}$$

The parameter vector $\theta_k$ is updated on a "slower" timescale. Two update rules can be obtained, based on two slightly different calculations:

$$\theta_{k+1} = \theta_k + \alpha_k(\phi_k - \gamma\phi'_k)(\phi_k^\top w_k) \quad \text{(an algorithm called GTD2), or} \tag{6}$$

$$\theta_{k+1} = \theta_k + \alpha_k\delta_k\phi_k - \alpha_k\gamma\phi'_k(\phi_k^\top w_k) \quad \text{(an algorithm called TDC).} \tag{7}$$

# 4 Nonlinear Temporal Difference Learning

Our goal is to generalize this approach to the case in which $V_\theta$ is a smooth, nonlinear function approximator. The first step is to find a good objective function on which to do gradient descent. In the linear case, MSPBE was chosen as a projection of the Bellman error on a natural hyperplane–the subspace to which $V_\theta$ is restricted. However, in the nonlinear case, the value function is no longer restricted to a plane, but can move on a nonlinear surface. More precisely, assuming that $V_\theta$ is a differentiable function of $\theta$, $\mathcal{M} = \{V_\theta \in \mathbb{R}^{|S|} \,|\, \theta \in \mathbb{R}^n\}$ becomes a differentiable submanifold of $\mathbb{R}^{|S|}$. Projecting onto a nonlinear manifold is not computationally feasible; to get around this problem, we will assume that the parameter vector $\theta$ changes very little in one step (given that learning rates are usually small); in this case, the surface is locally close to linear, and we can project onto the tangent plane at the given point. We now detail this approach and show that this is indeed a good objective function.

The *tangent plane* $PM_\theta$ of $\mathcal{M}$ at $\theta$ is the hyperplane of $\mathbb{R}^{|S|}$ that *(i)* passes through $V_\theta$ and *(ii)* is orthogonal to the normal of $\mathcal{M}$ at $\theta$. The *tangent space* $TM_\theta$ is the translation of $PM_\theta$ to the origin. Note that $TM_\theta = \{\Phi_\theta a \,|\, a \in \mathbb{R}^n\}$, where $\Phi_\theta \in \mathbb{R}^{|S| \times n}$ is defined by $(\Phi_\theta)_{s,i} = \frac{\partial}{\partial \theta_i} V_\theta(s)$. Let $\Pi_\theta$ be the projection that projects vectors of $(\mathbb{R}^{|S|}, \|\cdot\|_D)$ to $TM_\theta$. If $\Phi_\theta^\top D \Phi_\theta$ is non-singular then $\Pi_\theta$ can be written as:

$$\Pi_\theta = \Phi_\theta (\Phi_\theta^\top D \Phi_\theta)^{-1} \Phi_\theta^\top D. \tag{8}$$

The objective function that we will optimize is:

$$J(\theta) \;=\; \| \Pi_\theta (TV_\theta - V_\theta) \|_D^2 . \tag{9}$$

This is a natural generalization of the objective function defined by (3), as the plane on which we project is parallel to the tangent plane at $\theta$. More precisely, let $\Upsilon_\theta$ be the projection to $PM_\theta$ and let $\Pi_\theta$ be the projection to $TM_\theta$. Because the two hyperplanes are parallel, for any $V \in \mathbb{R}^{|S|}$, $\Upsilon_\theta V - V_\theta = \Pi_\theta (V - V_\theta)$. In other words, projecting onto the tangent space gives exactly the same distance as projecting onto the tangent plane, while being mathematically more convenient. Fig. 1 illustrates visually this objective function.

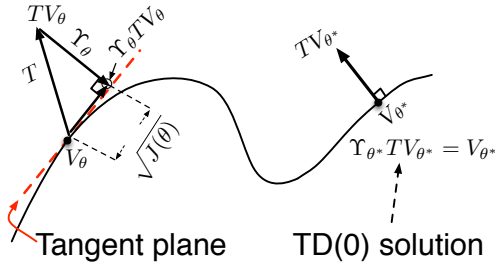

Figure 1: The MSPBE objective for nonlinear function approximation at two points in the value function space. The figure shows a point, $V_\theta$, at which, $J(\theta)$, is not 0 and a point, $V_{\theta^*}$, where $J(\theta^*) = 0$, thus $\Upsilon_{\theta^*} TV_{\theta^*} = V_{\theta^*}$, so this is a TD(0) solution.

We now show that $J(\theta)$ can be re-written in the same way as done in (Sutton et al., 2009b).

**Lemma 1.** *Assume $V_\theta(s_0)$ is continuously differentiable as a function of $\theta$, for any $s_0 \in \mathcal{S}$ s.t. $d(s_0) > 0$. Let $(s, \delta)$ be jointly distributed random variables as in Section 3 and assume that $\mathbb{E}[\nabla V_\theta(s) \nabla V_\theta(s)^\top]$ is nonsingular. Then*

$$J(\theta) = \mathbb{E}[\,\delta\, \nabla V_\theta(s)\,]^\top \, \mathbb{E}[\,\nabla V_\theta(s) \nabla V_\theta(s)^\top\,]^{-1} \, \mathbb{E}[\,\delta\, \nabla V_\theta(s)\,]. \tag{10}$$

*Proof.* The identity is obtained similarly to Sutton et. al (2009b), except that here $\Pi_\theta$ is expressed by (8). Details are omitted for brevity. $\qquad\square$

Note that the assumption that $\mathbb{E}[\,\nabla V_\theta(s) \nabla V_\theta(s)^\top\,]^{-1}$ is non-singular is akin to the assumption that the feature vectors are independent in the linear function approximation case. We make this assumption here for convenience; it can be lifted, but the proofs become more involved.

**Corollary 1.** *Under the conditions of Lemma 1, $J(\theta) = 0$, if and only if $V_\theta$ satisfies* (2).

This is an important corollary, because it shows that the global optima of the proposed objective function will not modify the set of solutions that the usual TD(0) algorithm would find (if it would indeed converge). We now proceed to compute the gradient of this objective.

**Theorem 1.** *Assume that* (i) $V_\theta(s_0)$ *is twice continuously differentiable in $\theta$ for any $s_0 \in S$ s.t. $d(s_0) > 0$ and* (ii) $W(\cdot)$ *defined by $W(\hat{\theta}) = \mathbb{E}[\nabla V_{\hat{\theta}} \nabla V_{\hat{\theta}}^\top]$ is non-singular in a small neighborhood of $\theta$. Let $(s, \delta)$ be jointly distributed random variables as in Section 3. Let $\phi \equiv \nabla V_\theta(s)$, $\phi' \equiv \nabla V_\theta(s')$ and*

$$h(\theta, u) = -\mathbb{E}[(\delta - \phi^\top u)\nabla^2 V_\theta(s)u],  \tag{11}$$

*where $u \in \mathbb{R}^n$. Then*

$$-\frac{1}{2}\nabla J(\theta) = -\mathbb{E}[(\gamma\phi' - \phi)\phi^\top w] + h(\theta, w) = -\mathbb{E}[\delta\phi] - \gamma\mathbb{E}[\phi'\phi^\top w] + h(\theta, w),  \tag{12}$$

*where $w = \mathbb{E}[\phi\phi^\top]^{-1}\mathbb{E}[\delta\phi]$.*

The main difference between Equation (12) and Equation (4), which shows the gradient for the linear case, is the appearance of the term $h(\theta, w)$, which involves second-order derivatives of $V_\theta$ (which are zero when $V_\theta$ is linear in $\theta$).

*Proof.* The conditions of Lemma 1 are satisfied, so (10) holds. Denote $\partial_i = \frac{\partial}{\partial\theta_i}$. From its definition and the assumptions, $W(u)$ is a symmetric, positive definite matrix, so $\frac{d}{du}(W^{-1})|_{u=\theta} = -W^{-1}(\theta)(\frac{d}{du}W|_{u=\theta})W^{-1}(\theta)$, where we use the assumption that $\frac{d}{du}W$ exists at $\theta$ and $W^{-1}$ exists in a small neighborhood of $\theta$. From this identity, we have:

$$
\begin{aligned}
-\frac{1}{2}[\nabla J(\theta)]_i &= -(\partial_i\mathbb{E}[\delta\phi])^\top\mathbb{E}[\phi\phi^\top]^{-1}\mathbb{E}[\delta\phi] - \frac{1}{2}\mathbb{E}[\delta\phi]^\top \partial_i(\mathbb{E}[\phi\phi^\top]^{-1})\mathbb{E}[\delta\phi] \\
&= -(\partial_i\mathbb{E}[\delta\phi])^\top\mathbb{E}[\phi\phi^\top]^{-1}\mathbb{E}[\delta\phi] + \frac{1}{2}\mathbb{E}[\delta\phi]^\top\mathbb{E}[\phi\phi^\top]^{-1}(\partial_i\mathbb{E}[\phi\phi^\top])\mathbb{E}[\phi\phi^\top]^{-1}\mathbb{E}[\delta\phi] \\
&= -\mathbb{E}[\partial_i(\delta\phi)]^\top(\mathbb{E}[\phi\phi^\top]^{-1}\mathbb{E}[\delta\phi]) + \frac{1}{2}(\mathbb{E}[\phi\phi^\top]^{-1}\mathbb{E}[\delta\phi])^\top\mathbb{E}[\partial_i(\phi\phi^\top)](\mathbb{E}[\phi\phi^\top]^{-1}\mathbb{E}[\delta\phi]).
\end{aligned}
$$

The interchange between the gradient and expectation is possible here because of assumptions *(i)* and *(ii)* and the fact that $S$ is finite. Now consider the identity

$$\frac{1}{2}x^\top\partial_i(\phi\phi^\top)x = \phi^\top x (\partial_i\phi^\top)x,$$

which holds for any vector $x \in \mathbb{R}^n$. Hence, using the definition of $w$,

$$
\begin{aligned}
-\frac{1}{2}[\nabla J(\theta)]_i &= -\mathbb{E}[\partial_i(\delta\phi)]^\top w + \frac{1}{2}w^\top\mathbb{E}[\partial_i(\phi\phi^\top)]w \\
&= -\mathbb{E}[(\partial_i\delta)\phi^\top w] - \mathbb{E}[\delta(\partial_i\phi^\top)w] + \mathbb{E}[\phi^\top w(\partial_i\phi^\top)w].
\end{aligned}
$$

Using $\nabla\delta = \gamma\phi' - \phi$ and $\nabla\phi^\top = \nabla^2 V_\theta(s)$, we get

$$-\frac{1}{2}\nabla J(\theta) = -\mathbb{E}[(\gamma\phi' - \hat{\phi})\phi^\top w] - \mathbb{E}[(\delta - \phi^\top w)\nabla^2 V(s)w],$$

Finally, observe that :

$$
\begin{aligned}
\mathbb{E}[(\gamma\phi' - \phi)\phi^\top w] &= \mathbb{E}[(\phi - \gamma\phi')\phi]^\top (\mathbb{E}[\phi\phi^\top]^{-1}\mathbb{E}[\delta\phi]) \\
&= \mathbb{E}[\delta\phi] - \mathbb{E}[\gamma\phi'\phi^\top](\mathbb{E}[\phi\phi^\top]^{-1}\mathbb{E}[\delta\phi]) = \mathbb{E}[\delta\phi] - \mathbb{E}[\gamma\phi'\phi^\top w].
\end{aligned}
$$

which concludes the proof. $\qquad\square$

Theorem 1 suggests straightforward generalizations of GTD2 and TDC (cf. Equations (6) and (7)) to the nonlinear case. Weight $w_k$ is updated as before on a "faster" timescale:

$$w_{k+1} = w_k + \beta_k(\delta_k - \phi_k^\top w_k)\phi_k.  \tag{13}$$

The parameter vector $\theta_k$ is updated on a "slower" timescale, either according to

$$\theta_{k+1} = \Gamma\left(\theta_k + \alpha_k\left\{(\phi_k - \gamma\phi_k')(\phi_k^\top w_k) - h_k\right\}\right), \qquad \text{(non-linear GTD2)}  \tag{14}$$

or, according to

$$\theta_{k+1} = \Gamma\Big(\theta_k + \alpha_k\left\{\delta_k\phi_k - \gamma\phi'_k(\phi_k^\top w_k) - h_k\right\}\Big), \qquad \text{(non-linear TDC)} \tag{15}$$

where

$$h_k = (\delta_k - \phi_k^\top w_k)\,\nabla^2 V_{\theta_k}(s_k)w_k. \tag{16}$$

Besides $h_k$, the only new ingredient compared to the linear case is $\Gamma : \mathbb{R}^n \to \mathbb{R}^n$, a mapping that projects its argument into an appropriately chosen compact set $C$ with a smooth boundary. The purpose of this projection is to prevent the parameters to diverge in the initial phase of the algorithm, which could happen due to the presence of the nonlinearities in the algorithm. Projection is a common technique for stabilizing the transient behavior of stochastic approximation algorithms (see, e.g., Kushner & Yin, 2003). In practice, if one selects $C$ large enough so that it contains the set of possible solutions $U = \{\,\theta\,|\,\mathbb{E}[\delta\,\nabla V_\theta(s)] = 0\,\}$ (by using known bounds on the size of the rewards and on the derivative of the value function), it is very likely that no projections will take place at all during the execution of the algorithm. We expect this to happen frequently in practice: the main reason for the projection is to facilitate convergence analysis.

Let us now analyze the computational complexity per update. Assume that $V_\theta(s)$ and its gradient can each be computed in $O(n)$ time, the usual case for approximators of interest (e.g., neural networks). Equation (16) also requires computing the product of the Hessian of $V_\theta(s)$ and $w$. Pearlmutter (1994) showed that this can be computed exactly in $O(n)$ time. The key is to note that $\nabla^2 V_{\theta_k}(s_k)w_k = \nabla(\nabla V_{\theta_k}(s)^\top w_k)$, because $w_k$ does not depend on $\theta_k$. The scalar term $\nabla V_{\theta_k}(s)^\top w_k$ can be computed in $O(n)$ and its gradient, which is a vector, can also be computed in $O(n)$. Hence, the computation time per update for the proposed algorithms is linear in the number of parameters of the function approximator (just like in TD(0)).

## 5 Convergence Analysis

Given the compact set $C \subset \mathbb{R}^n$, let $\mathcal{C}(C)$ be the space of $C \to \mathbb{R}^n$ continuous functions. Given projection $\Gamma$ onto $C$, let operator $\hat{\Gamma} : \mathcal{C}(C) \to \mathcal{C}(\mathbb{R}^n)$ be

$$\hat{\Gamma}v\,(\theta) = \lim_{0 < \varepsilon \to 0} \frac{\Gamma\big(\theta + \varepsilon\,v(\theta)\big) - \theta}{\varepsilon}.$$

By assumption, $\Gamma(\theta) = \arg\min_{\theta' \in C}\|\theta' - \theta\|$ and the boundary of $C$ is smooth, so $\hat{\Gamma}$ is well defined. In particular, $\hat{\Gamma}v\,(\theta) = v(\theta)$ when $\theta \in C^\circ$, otherwise, if $\theta \in \partial C$, $\hat{\Gamma}v\,(\theta)$ is the projection of $v(\theta)$ to the tangent space of $\partial C$ at $\theta$. Consider the following ODE:

$$\dot{\theta} = \hat{\Gamma}(-\tfrac{1}{2}\nabla J)(\theta), \quad \theta(0) \in C. \tag{17}$$

Let $K$ be the set of all asymptotically stable equilibria of (17). By the definitions, $K \subset C$. Furthermore, $U \cap C \subset K$.

The next theorem shows that under some technical conditions, the iterates produced by nonlinear GTD2 converge to $K$ with probability one.

**Theorem 2** (Convergence of nonlinear GTD2). *Let $(s_k, r_k, s'_k)_{k\geq 0}$ be a sequence of transitions that satisfies A1. Consider the nonlinear GTD2 updates (13), (14). with positive step-size sequences that satisfy $\sum_{k=0}^\infty \alpha_k = \sum_{k=0}^\infty \beta_k = \infty$, $\sum_{k=0}^\infty \alpha_k^2$, $\sum_{k=0}^\infty \beta_k^2 < \infty$ and $\frac{\alpha_k}{\beta_k} \to 0$, as $k \to \infty$. Assume that for any $\theta \in C$ and $s_0 \in \mathcal{S}$ s.t. $d(s_0) > 0$, $V_\theta(s_0)$ is three times continuously differentiable. Further assume that for each $\theta \in C$, $\mathbb{E}[\phi_\theta\phi_\theta^\top]$ is nonsingular. Then $\theta_k \to K$, with probability one, as $k \to \infty$.*

*Proof.* Let $(s, r, s')$ be a random transition. Let $\phi_\theta = \nabla V_\theta(s)$, $\phi'_\theta = \nabla V_\theta(s')$, $\phi_k = \nabla V_{\theta_k}(s_k)$, and $\phi'_k = \nabla V_{\theta_k}(s'_k)$. We begin by rewriting the updates (13)-(14) as follows:

$$w_{k+1} = w_k + \beta_k(f(\theta_k, w_k) + M_{k+1}), \tag{18}$$

$$\theta_{k+1} = \Gamma\big(\theta_k + \alpha_k(g(\theta_k, w_k) + N_{k+1})\big), \tag{19}$$

where

$$
\begin{aligned}
f(\theta_k, w_k) &= \mathbb{E}[\delta_k\phi_k|\theta_k] - \mathbb{E}[\phi_k\phi_k^\top|\theta_k]w_k, & M_{k+1} &= (\delta_k - \phi_k^\top w_k)\phi_k - f(\theta_k, w_k), \\
g(\theta_k, w_k) &= \mathbb{E}\big[(\phi_k - \gamma\phi'_k)\phi_k^\top w_k - h_k|\theta_k, w_k\big], & N_{k+1} &= ((\phi_k - \gamma\phi'_k)\phi_k^\top w_k - h_k) - g(\theta_k, w_k).
\end{aligned}
$$

We need to verify that there exists a compact set $B \subset \mathbb{R}^{2n}$ such that *(a)* the functions $f(\theta, w)$, $g(\theta, w)$ are Lipschitz continuous over $B$, *(b)* $(M_k, \mathcal{G}_k)$, $(N_k, \mathcal{G}_k)$, $k \geq 1$ are martingale difference sequences, where $\mathcal{G}_k = \sigma(r_i, \theta_i, w_i, s_i, i \leq k; s_i', i < k)$, $k \geq 1$ are increasing sigma fields, *(c)* $\{(w_k(\theta), \theta)\}$ with $w_k(\theta)$ obtained as $\delta_k(\theta) = r_k + \gamma V_\theta(s_k') - V_\theta(s_k)$, $\phi_k(\theta) = \nabla V_\theta(s_k)$,

$$w_{k+1}(\theta) = w_k(\theta) + \beta_k \Big( \delta_k(\theta) - \phi_k(\theta)^\top w_k(\theta) \Big) \phi_k(\theta)$$

almost surely stays in $B$ for any choice of $(w_0(\theta), \theta) \in B$, and *(d)* $\{(w, \theta_k)\}$ almost surely stays in $B$ for any choice of $(w, \theta_0) \in B$. From these and the conditions on the step-sizes, using standard arguments (c.f. Theorem 2 of Sutton et al. (2009b)), it follows that $\theta_k$ converges almost surely to the set of asymptotically stable equilibria of $\dot{\theta} = \hat{\Gamma} F(\theta)$, $(\theta(0) \in C)$, where $F(\theta) = g(\theta, w_\theta)$. Here for $\theta \in C$ fixed, $w_\theta$ is the (unique) equilibrium point of

$$\dot{w} = \mathbb{E}[\delta_\theta \phi_\theta] - \mathbb{E}[\phi_\theta \phi_\theta^\top] w, \tag{20}$$

where $\delta_\theta = r + \gamma V_\theta(s') - V_\theta(s)$. Clearly, $w_\theta = \mathbb{E}\big[\phi_\theta \phi_\theta^\top\big]^{-1} \mathbb{E}[\delta_\theta \phi_\theta]$, which exists by assumption. Then by Theorem 1 it follows that $F(\theta) = -\frac{1}{2} \nabla J(\theta)$. Hence, the statement will follow once (a)–(d) are verified.

Note that (a) is satisfied because $V_\theta$ is three times continuously differentiable. For (b), we need to verify that for any $k \geq 0$, $\mathbb{E}[M_{k+1} \mid \mathcal{G}_k] = 0$ and $\mathbb{E}[N_{k+1} \mid \mathcal{G}_k] = 0$, which in fact follow from the definitions. Condition (c) follows since, by a standard argument (e.g., Borkar & Meyn, 2000), $w_k(\theta)$ converges to $w_\theta$, which by assumption stays bounded if $\theta$ comes from a bounded set. For condition (d), note that $\{\theta_k\}$ is uniformly bounded since for any $k \geq 0$, $\theta_k \in C$, and by assumption $C$ is a compact set. $\qquad \square$

**Theorem 3** (Convergence of nonlinear TDC). *Under the same conditions as in Theorem 2, the iterates computed via* (13), (15) *satisfy* $\theta_k \to K$, *with probability one, as* $k \to \infty$.

The proof follows in a similar manner as that of Theorem 2 and is omitted for brevity.

## 6 Empirical results

To illustrate the convergence properties of the algorithms, we applied them to the "spiral" counterexample of Tsitsiklis & Van Roy (1997), originally used to show the divergence of TD(0) with nonlinear function approximation. The Markov chain with 3 states is shown in the left panel of Figure 2. The reward is always zero and the discount factor is $\gamma = 0.9$. The value function has a single parameter, $\theta$, and takes the nonlinear spiral form $V_\theta(s) = \Big( a(s) \cos(\hat{\lambda}\theta) - b(s) \sin(\hat{\lambda}\theta) \Big) e^{\epsilon\theta}$. The true value function is $V = (0, 0, 0)^\top$ which is achieved as $\theta \to -\infty$. Here we used $V_0 = (100, -70, -30)^\top$, $a = V_0$, $b = (23.094, -98.15, 75.056)^\top$, $\hat{\lambda} = 0.866$ and $\epsilon = 0.05$. Note that this is a degenerate example, in which our theorems do not apply, because the optimal parameter values are infinite. Hence, we run our algorithms without a projection step. We also use constant learning rates, in order to facilitate gradient descent through an error surface which is essentially flat. For TDC we used $\alpha = 0.5$, $\beta = 0.05$, and for GTD2, $\alpha = 0.8$ and $\beta = 0.1$. For TD(0) we used $\alpha = 2 \times 10^{-3}$ (as argued by Tsitsiklis & Van Roy (1997), tuning the step-size does not help with the divergence problem). All step sizes are then normalized by $\|V_\theta^\top D \frac{d}{d\theta} V_\theta\|$. The graph shows the performance measure, $\sqrt{J}$, as a function of the number of updates (we used expected updates for all algorithms). GTD2 and TDC converge to the correct solution, while TD(0) diverges. We note that convergence happens despite the fact that this example is outside the scope of the theory.

To assess the performance of the new algorithms on a large scale problem, we used them to learn an evaluation function in 9x9 computer Go. We used a version of RLGO (Silver, 2009), in which a logistic function is fit to evaluate the probability of winning from a given position. Positions were described using 969,894 binary features corresponding to all possible shapes in every 3x3, 2x2, and 1x1 region of the board. Using weight sharing to take advantage of symmetries, the million features were reduced to a parameter vector of $n = 63,303$ components. Experience was generated by self-play, with actions chosen uniformly randomly among the legal moves. All rewards were zero, except upon winning the game, when the reward was 1. We applied four algorithms to this problem: TD(0), the proposed algorithms (GTD2 and TDC) and residual gradient (RG). In the experiments, RG was

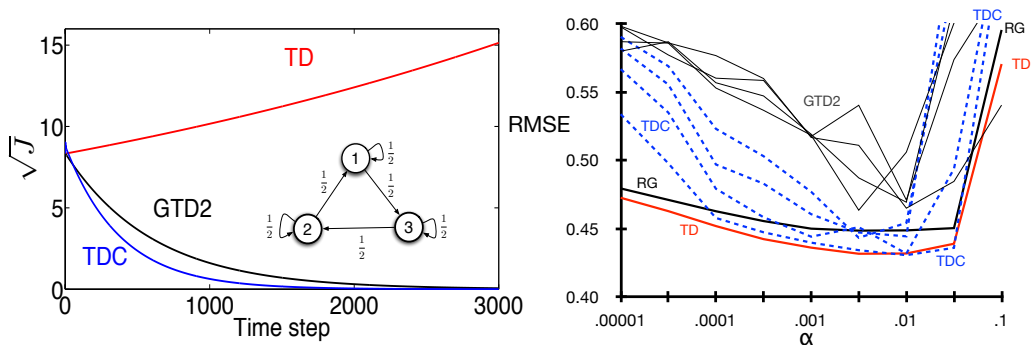

Figure 2: Empirical evaluation results. Left panel: example MDP from Tsitsiklis & Van Roy (1994). Right panel: 9x9 Computer Go.

run with only one sample[3]. In each run, $\theta$ was initialized to random values uniformly distributed in $[-0.1, 0.1]$; for GTD2 and TDC, the second parameter vector, $w$, was initialized to 0. Training then proceeded for 5000 complete games, after which $\theta$ was frozen. This problem is too large to compute the objective function $J$. Instead, to assess the quality of the solutions obtained, we estimated the average prediction error of each algorithm. More precisely, we generated 2500 test games; for every state occurring in a game, we computed the squared error between its predicted value and the actual return that was obtained in that game. We then computed the root of the mean-squared error, averaged over all time steps. The right panel in Figure 2 plots this measure over a range of values of the learning rate $\alpha$. The results are averages over 50 independent runs. For TDC and GTD we used several values of the $\beta$ parameter, which generate the different curves. As was noted in previous empirical work, TD provides slightly better estimates than the RG algorithm. TDC's performance is very similar to TD, for a wide range of parameter values. GTD2 is slightly worse. These results are very similar in flavor to those obtained in Sutton et al. (2009b) using the same domain, but with linear function approximation.

## 7 Conclusions and future work

In this paper, we solved a long-standing open problem in reinforcement learning, by establishing a family of temporal-difference learning algorithms that converge with arbitrary differentiable function approximators (including neural networks). The algorithms perform gradient descent on a natural objective function, the projected Bellman error. The local optima of this function coincide with solutions that could be obtained by TD(0). Of course, TD(0) need not converge with non-linear function approximation. Our algorithms are on-line, incremental and their computational cost per update is linear in the number of parameters. Our theoretical results guarantee convergence to a local optimum, under standard technical assumptions. Local optimality is the best one can hope for, since nonlinear function approximation creates non-convex optimization problems. The early empirical results obtained for computer Go are very promising. However, more practical experience with these algorithms is needed. We are currently working on extensions of these algorithms using eligibility traces, and on using them for solving control problems.

**Acknowledgments**

This research was supported in part by NSERC, iCore, AICML and AIF. We thank the three anonymous reviewers for their useful comments on previous drafts of this paper.

**References**

Antos, A., Szepesvári, Cs. & Munos, R. (2008). Learning near-optimal policies with Bellman-residual minimization based fitted policy iteration and a single sample path. *Machine Learning 71*: 89–129.

Baird, L. C. (1995). Residual algorithms: Reinforcement learning with function approximation. In *Proceedings of the Twelfth International Conference on Machine Learning*, pp. 30–37. Morgan Kaufmann.

Borkar, V. S. & Meyn, S. P. (2000). The ODE method for convergence of stochastic approximation and reinforcement learning. *SIAM Journal on Control And Optimization 38(2)*: 447–469.

Boyan, J. A. & Moore, A.W. (1995). Generalization in Reinforcement Learning: Safely Approximating the Value Function. In *Advances in Neural Information Processing Systems 7*, pp. 369–376, MIT Press.

Crites, R. H. & Barto, A.G. (1995). Improving Elevator Performance Using Reinforcement Learning In *Advances in Neural Information Processing Systems 8*, pp. 1017-1023. MIT Press.

Farahmand, A.m., Ghavamzadeh, M., Szepesvari, C. & Mannor, S. (2009). Regularized Policy Iteration In *Advances in Neural Information Processing Systems 21*, pp. 441–448.

Kushner, H. J. & Yin, G. G. (2003). *Stochastic Approximation Algorithms and Applications*. Second Edition, Springer-Verlag.

Pearlmutter, B. A (1994). Fast exact multiplication by the Hessian. *Neural Computation 6(1)*, pp. 147–160.

Silver, D. (2009). *Reinforcement Learning and Simulation-Based Search in Computer Go*. University of Alberta Ph.D. thesis.

Sutton, R. S. (1988). Learning to predict by the method of temporal differences. *Machine Learning 3*:9–44.

Sutton, R. S. & Barto, A. G. (1998). *Reinforcement Learning: An Introduction*. MIT Press.

Sutton, R. S., Szepesvári, Cs. & Maei, H. R. (2009a). A convergent $O(n)$ algorithm for off-policy temporal-difference learning with linear function approximation. In *Advances in Neural Information Processing Systems 21*, pp. 1609–1616. MIT Press.

Sutton, R. S., Maei, H. R, Precup, D., Bhatnagar, S., Silver, D., Szepesvári, Cs. & Wiewiora, E. (2009b). Fast gradient-descent methods for temporal-difference learning with linear function approximation. In *Proceedings of the 26th International Conference on Machine Learning*, pp. 993–1000. Omnipress.

Tesauro, G. (1992) Practical issues in temporal difference learning. *Machine Learning 8*: 257-277.

Tsitsiklis, J. N. & Van Roy, B. (1997). An analysis of temporal-difference learning with function approximation. *IEEE Transactions on Automatic Control 42*:674–690.

Zhang, W. & Dietterich, T. G. (1995) A reinforcement learning approach to job-shop scheduling. In *Proceedings of the Fourteenth International Joint Conference on Artificial Intelligence*, pp. 1114-1120. AAAI Press.

## Footnotes

[1] Under appropriate technical conditions, our results, can be generalized to MDPs with infinite state spaces, but we do not address this here.

[2]This error function was also described in (Antos et al., 2008), although the algorithmic issue of how to minimize it is not pursued there. Algorithmic issues in a batch setting are considered by Farahmand et al. (2009) who also study regularization.

[3]Unlike TD, RG would require two independent transition samples from a given state. This requires knowledge about the model of the environment which is not always available. In the experiments only one transition sample was used following Baird's original recommendation.
